# Nonstandard Interpretations of Probabilistic Programs for Efficient Inference

**David Wingate**
BCS / LIDS, MIT
wingated@mit.edu

**Noah D. Goodman**
Psychology, Stanford
ngoodman@stanford.edu

**Andreas Stuhlmüller**
BCS, MIT
ast@mit.edu

**Jeffrey M. Siskind**
ECE, Purdue
qobi@purdue.edu

## Abstract

Probabilistic programming languages allow modelers to specify a stochastic process using syntax that resembles modern programming languages. Because the program is in machine-readable format, a variety of techniques from compiler design and program analysis can be used to examine the structure of the distribution represented by the probabilistic program. We show how *nonstandard interpretations* of probabilistic programs can be used to craft efficient inference algorithms: information about the structure of a distribution (such as gradients or dependencies) is generated as a monad-like side computation while executing the program. These interpretations can be easily coded using special-purpose objects and operator overloading. We implement two examples of nonstandard interpretations in two different languages, and use them as building blocks to construct inference algorithms: automatic differentiation, which enables gradient based methods, and provenance tracking, which enables efficient construction of global proposals.

## 1 Introduction

Probabilistic programming simplifies the development of probabilistic models by allowing modelers to specify a stochastic process using syntax that resembles modern programming languages. These languages permit arbitrary mixing of deterministic and stochastic elements, resulting in tremendous modeling flexibility. The resulting programs define probabilistic models that serve as prior distributions: running the (unconditional) program forward many times results in a distribution over execution traces, with each trace being a sample from the prior. Examples include BLOG [13], Bayesian Logic Programs [10] IBAL[18], CHURCH [6], Stochastic MATLAB [28], and HANSEI [11].

The primary challenge in developing such languages is scalable inference. Inference can be viewed as reasoning about the posterior distribution over execution traces conditioned on a particular program output, and is difficult because of the flexibility these languages present: in principle, an inference algorithm must behave reasonably for any program a user wishes to write. Sample-based MCMC algorithms are the state-of-the-art method, due to their simplicity, universality, and compositionality. But in probabilistic modeling more generally, efficient inference algorithms are designed by taking advantage of structure in distributions. How can we find structure in a distribution defined by a probabilistic program? A key observation is that some languages, such as CHURCH and Stochastic MATLAB, are defined in terms of an existing (non-probabilistic) language. Programs in these languages may literally be executed in their native environments—suggesting that tools from program analysis and programming language theory can be leveraged to find and exploit structure in the program for inference, much as a compiler might find and exploit structure for performance.

Here, we show how *nonstandard interpretations* of probabilistic programs can help craft efficient inference algorithms. Information about the structure of a distribution (such as gradients, dependencies or bounds) is generated as a monad-like side computation while executing the program. This extra information can be used to, for example, construct good MH proposals, or search efficiently for a local maximum. We focus on two such interpretations: automatic differentiation and provenance tracking, and show how they can be used as building blocks to construct efficient inference

algorithms. We implement nonstandard interpretations in two different languages (CHURCH and Stochastic MATLAB), and experimentally demonstrate that while they typically incur some additional execution overhead, they dramatically improve inference performance.

## 2 Background and Related Work

We begin by outlining our setup, following [28]. We define an unconditioned probabilistic program to be a parameterless function $f$ with an arbitrary mix of stochastic and deterministic elements (hereafter, we will use the term function and program interchangeably). The function $f$ may be written in any language, but our running example will be MATLAB. We allow the function to be arbitrarily complex inside, using any additional functions,

| **Alg. 1: A Gaussian-Gamma mixture** |
|---|
| 1: for i=1:1000 |
| 2:   if ( rand > 0.5 ) |
| 3:     X(i) = randn; |
| 4:   else |
| 5:     X(i) = gammarnd; |
| 6:   end; |
| 7: end; |

recursion, language constructs or external libraries it wishes. The only constraint is that the function must be self-contained, with no external side-effects which would impact the execution of the function from one run to another.

The stochastic elements of $f$ must come from a set of known, fixed *elementary random primitives*, or ERPs. Complex distributions are constructed compositionally, using ERPs as building blocks. In MATLAB, ERPs may be functions such as rand (sample uniformly from [0,1]) or randn (sample from a standard normal). Higher-order random primitives, such as nonparametric distributions, may also be defined, but must be fixed ahead of time. Formally, let $\mathcal{T}$ be the set of ERP types. We assume that each type $t \in \mathcal{T}$ is a parametric family of distributions $p_t(x|\theta_t)$, with parameters $\theta_t$.

Now, consider what happens while executing $f$. As $f$ is executed, it encounters a series of ERPs. Alg. 1 shows an example of a simple $f$ written in MATLAB with three syntactic ERPs: rand, randn, and gammarnd. During execution, depending on the return value of each call to rand, different paths will be taken through the program, and different ERPs will be encountered. We call this path an *execution trace*. A total of 2000 random choices will be made when executing this $f$.

Let $f_{k|x_1,\cdots,x_{k-1}}$ be the $k$'th ERP encountered while executing $f$, and let $x_k$ be the value it returns. Note that the parameters passed to the $k$'th ERP may change depending on previous $x_k$'s (indeed, its type may also change, as well as the total number of ERPs). We denote by $x$ all of the random choices which are made by $f$, so $f$ defines the probability distribution $p(x)$. In our example, $x \in \mathbb{R}^{2000}$. The probability $p(x)$ is the product of the probability of each individual ERP choice:

$$p(x) = \prod_{k=1}^{K} p_{t_k}(x_k|\theta_{t_k}, x_1, \cdots, x_{k-1}) \tag{1}$$

again noting explicitly that types and parameters may depend arbitrarily on previous random choices. To simplify notation, we will omit the conditioning on the values of previous ERPs, but again wish to emphasize that these dependencies are critical and cannot be ignored. By $f_k$, it should therefore be understood that we mean $f_{k|x_1,\cdots,x_{k-1}}$, and by $p_{t_k}(x_k|\theta_{t_k})$ we mean $p_{t_k}(x_k|\theta_{t_k}, x_1, \cdots, x_{k-1})$.

Generative functions as described above are, of course, easy to write. A much harder problem, and our goal in this paper, is to reason about the posterior conditional distribution $p(x|y)$, where we define $y$ to be a subset of random choices which we condition on and (in an abuse of notation) $x$ to be the remaining random choices. For example, we may condition $f$ on the X(i)'s, and reason about the sequence of rand's most likely to generate the X(i)'s. For the rest of this paper, we will drop $y$ and simply refer to $p(x)$, but it should be understood that the goal is always to perform inference in conditional distributions.

### 2.1 Nonstandard Interpretations of Probabilistic Programs

With an outline of probabilistic programming in hand, we now turn to nonstandard interpretations. The idea of nonstandard interpretations originated in model theory and mathematical logic, where it was proposed that a set of axioms could be interpreted by different models. For example, differential geometry can be considered a nonstandard interpretation of classical arithmetic.

In programming, a nonstandard interpretation replaces the domain of the variables in the program with a new domain, and redefines the semantics of the operators in the program to be consistent with the new domain. This allows reuse of program syntax while implementing new functionality. For example, the expression "$a * b$" can be interpreted equally well if $a$ and $b$ are either scalars or

matrices, but the "∗" operator takes on different meanings. Practically, many useful nonstandard interpretations can be implemented with operator overloading: variables are redefined to be objects with operators that implement special functionality, such as tracing, reference counting, or profiling.

For the purposes of inference in probabilistic programs, we will augment each random choice $x_k$ with additional side information $s_k$, and replace each $x_k$ with the tuple $\langle x_k, s_k \rangle$. The native interpreter for the probabilistic program can then interpret the source code as a sequence of operations on these augmented data types. For a recent example of this, we refer the reader to [24].

## 3 Automatic Differentiation

For probabilistic models with many continuous-valued random variables, the gradient of the likelihood $\nabla_x p(x)$ provides local information that can significantly improve the properties of Monte-Carlo inference algorithms. For instance, Langevin Monte-Carlo [20] and Hamiltonian MCMC [15] use this gradient as part of a variable-augmentation technique (described below). We would like to be able to use gradients in the probabilistic-program setting, but $p(x)$ is represented implicitly by the program. How can we compute its gradient? We use *automatic differentiation* (AD) [3, 7], a nonstandard interpretation that automatically constructs $\nabla_x p(x)$. The automatic nature of AD is critical because it relieves the programmer from hand-computing derivatives for each model; moreover, some probabilistic programs dynamically create or delete random variables making simple closed-form expressions for the gradient very difficult to find.

Unlike finite differencing, AD computes an exact derivative of a function $f$ at a point (up to machine precision). To do this, AD relies on the chain rule to decompose the derivative of $f$ into derivatives of its sub-functions: ultimately, known derivatives of elementary functions are composed together to yield the derivative of the compound function. This composition can be computed as a nonstandard interpretation of the underlying elementary functions.

The derivative computation as a composition of the derivatives of the elementary functions can be performed in different orders. In *forward mode* AD [27], computation of the derivative proceeds by propagating perturbations of the input toward the output. This can be done by a nonstandard interpretation that extends each real value to the first two terms of its Taylor expansion [26], overloading each elementary function to operate on these real "polynomials". Because the derivatives of $f$ at $c$ can be extracted from the coefficients of $\epsilon$ in $f(c + \epsilon)$, this allows computation of the gradient. In *reverse mode* AD [25], computation of the derivative proceeds by propagating sensitivities of the output toward the input. One way this can be done is by a nonstandard interpretation that extends each real value into a "tape" that captures the trace of the real computation which led to that value from the inputs, overloading each elementary function to incrementally construct these tapes. Such a tape can be postprocessed, in a fashion analogous to backpropagation [21], to yield the gradient. These two approaches have complementary computational tradeoffs: reverse mode (which we use in our implementation) can compute the gradient of a function $f : \mathbb{R}^n \to \mathbb{R}$ with the same asymptotic time complexity as computing $f$, but not the same asymptotic space complexity (due to its need for saving the computation trace), while forward mode can compute the gradient with these same asymptotic space complexity, but with a factor of $O(n)$ slowdown (due to its need for constructing the gradient out of partial derivatives along each independent variable).

There are implementations of AD for many languages, including SCHEME(e.g., [17]), FORTRAN (e.g., ADIFOR[2]), C (e.g., ADOL−C [8]), C++ (e.g., FADBAD++[1]), MATLAB (e.g., INTLAB [22]), and MAPLE (e.g., GRADIENT [14]). See www.autodiff.org. Additionally, overloading and AD are well established techniques that have been applied to machine learning, and even to application-specific programming languages for machine learning, e.g., LUSH[12] and DYNA[4]. In particular, DYNA applies a nonstandard interpretation for $\wedge$ and $\vee$ as a semiring ($\times$ and $+$, $+$ and $\max$, ...) in a memoizing PROLOG to generalize Viterbi, forward/backward, inside/outside, etc. and uses AD to derive the outside algorithm from the inside algorithm and support parameter estimation, but unlike probabilistic programming, it does not model general stochastic processes and does not do general inference over such. Our use of overloading and AD differs in that it facilitates inference in complicated models of general stochastic processes formulated as probabilistic programs. Probabilistic programming provides a powerful and convenient framework for formulating complicated models and, more importantly, separating such models from orthogonal inference mechanisms. Moreover, overloading provides a convenient mechanism for implementing many such inference mechanisms (e.g., Langevin MC, Hamiltonian MCMC, Provenance Tracking, as demonstrated below) in a probabilistic programming language.

```
(define (perlin-pt x y keypt power)
  (* 255 (sum (map (lambda (p2 pow)
    (let ((x0 (floor (* p2 x))) (y0 (floor (* p2 y))))
      (* pow (2d-interp (keypt x0 y0) (keypt (+ 1 x0) y0) (keypt x0 (+ 1 y0)) (keypt (+ 1 x0) (+ 1 y0))))))
        powers-of-2 power))))

(define (perlin xs ys power)
  (let ([keypt (mem (lambda (x y) (/ 1 (+ 1 (exp (- (gaussian 0.0 2.0)))))))]
    (map (lambda (x) (map (lambda (y) (perlin-pt x y keypt power)) xs)) ys)))
```

Figure 1: Code for the structured Perlin noise generator. `2d-interp` is B-spline interpolation.

## 3.1 Hamiltonian MCMC

To illustrate the power of AD in probabilistic programming, we build on Hamiltonian MCMC (HMC), an efficient algorithm whose popularity has been somewhat limited by the necessity of computing gradients—a difficult task for complex models. Neal [15] introduces HMC as an inference method which "produces distant proposals for the Metropolis algorithm, thereby avoiding the slow exploration of the state space that results from the diffusive behavior of simple random-walk proposals."

| **Alg. 2: Hamiltonian MCMC** |
| --- |
| 1: repeat forever |
| 2:   **Gibbs step:** |
| 3:   Draw momentum $m \sim \mathcal{N}(0, \sigma^2)$ |
| 4:   **Metropolis step:** |
| 5:   Start with current state $(x, m)$ |
| 6:   Simulate Hamiltonian dynamics to give $(x', m')$ |
| 7:   Accept w/ $p = \min[1, e^{(-H(x',m')+H(x,m))}]$ |
| 8: end; |

HMC begins by augmenting the states space with "momentum variables" $m$. The distribution over this augmented space is $e^{H(x,m)}$, where the Hamiltonian function $H$ decomposed into the sum of a potential energy term $U(x) = -\ln p(x)$ and a kinetic energy $K(m)$ which is usually taken to be Gaussian. Inference proceeds by alternating between a Gibbs step and Metropolis step: fixing the current state $x$, a new momentum $m$ is sampled from the prior over $m$; then $x$ and $m$ are updated together by following a trajectory according to Hamiltonian dynamics. Discrete integration of Hamiltonian dynamics requires the gradient of $H$, and must be done with a symplectic (i.e. volume preserving) integrator (following [15] we use the Leapfrog method). While this is a complex computation, incorporating gradient information dramatically improves performance over vanilla random-walk style MH moves (such as Gaussian drift kernels), and its statistical efficiency also scales much better with dimensionality than simpler methods [15]. AD can also compute higher-order derivatives. For example, Hessian matrices can be used to construct blocked Metropolis moves [9] or proposals based on Newton's method [19], or as part of Riemannian manifold methods [5].

## 3.2 Experiments and Results

We implemented HMC by extending BHER [28], a lightweight implementation of the CHURCH language which provides simple, but universal, MH inference. We used used an implementation of AD based on [17] that uses hygienic operator overloading to do both forward and reverse mode AD for Scheme (the target language of the BHER compiler).

The goal is to compute $\nabla_x p(x)$. By Eq. 1, $p(x)$ is the product of the individual choices made by each $x_i$ (though each probability can depend on previous choices, through the program evaluation). To compute $p(x)$, BHER executes the corresponding program, accumulating likelihoods. Each time a continuous ERP is created or retrieved, we wrap it in a "tape" object which is used to track gradient information; as the likelihood $p(x)$ is computed, these tapes flow through the program and through appropriately overloaded operators, resulting in a dependency graph for the real portion of the computation. The gradient is then computed in reverse mode, by "back-propagating" along this graph. We implement an HMC kernel by using this gradient in the leapfrog integrator. Since program states may contain a combination of discrete and continuous ERPs, we use an overall cycle kernel which alternates between standard MH kernel for individual discrete random variables and the HMC kernel for all continuous random choices. To decrease burn-in time, we initialize the sampler by using annealed gradient ascent (again implemented using AD).

We ran two sets of experiments that illustrate two different benefits of HMC with AD: automated gradients of complex code, and good statistical efficiency.

**Structured Perlin noise generation.** Our first experiment uses HMC to generate modified Perlin noise with soft symmetry structure. Perlin noise is a procedural texture used by computer graphics artists to add realism to natural textures such as clouds, grass or tree bark. We generate Perlin-like noise by layering octaves of random but smoothly varying functions. We condition the result

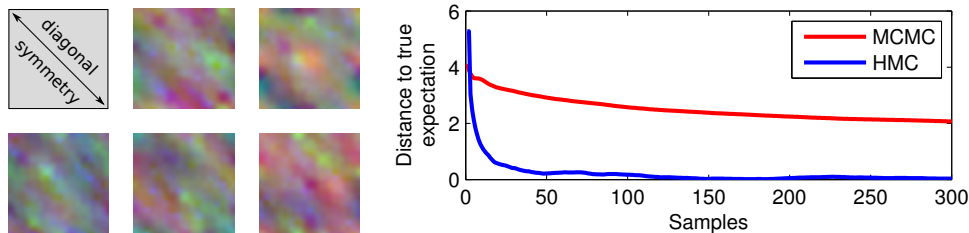

Figure 2: On the left: samples from the structured Perlin noise generator. On the right: convergence of expected mean for a draw from a 3D spherical Gaussian conditioned on lying on a line.

on approximate diagonal symmetry, forcing the resulting image to incorporate additional structure without otherwise skewing the statistics of the image. Note that the MAP solution for this problem is uninteresting, as it is a uniform image; it is the variations around the MAP that provide rich texture. We generated 48x48 images; the model had roughly 1000 variables.

Fig. 2 shows the result via typical samples generated by HMC, where the approximate symmetry is clearly visible. A code snippet demonstrating the complexity of the calculations is shown in Fig. 1; this experiment illustrates how the automatic nature of the gradients is most helpful, as it would be time consuming to compute these gradients by hand—particularly since we are free to condition using any function of the image.

**Complex conditioning.** For our second example, we demonstrate the improved statistical efficiency of the samples generated by HMC versus BHER's standard MCMC algorithm. The goal is to sample points from a complex 3D distribution, defined by starting with a Gaussian prior, and sampling points that are noisily conditioned to be on a line running through $\mathbb{R}^3$. This creates complex interactions with the prior to yield a smooth, but strongly coupled, energy landscape.

Fig. 2 compares our HMC implementation with BHER's standard MCMC engine. The x-axis denotes samples, while the y-axis denotes the convergence of an estimator of certain marginal statistics of the samples. We see that this estimator converges much faster for HMC, implying

**Normal distribution noisily conditioned on line (2D projection)**

1: $x \sim \mathcal{N}(\mu, \sigma)$
2: $k \sim \text{Bernoulli}(e^{-\frac{\text{dist}(\text{line}, x)}{\text{noise}}})$
3: Condition on $k = 1$

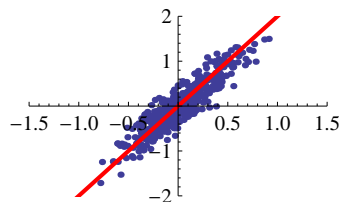

that the samples which are generated are less autocorrelated – affirming that HMC is indeed making better distal moves. HMC is about 5x slower than MCMC for this experiment, but the overhead is justified by the significant improvement in the statistical quality of the samples.

## 4   Provenance Tracking for Fine-Grained Dynamic Dependency Analysis

One reason gradient based inference algorithms are effective is that the chain rule of derivatives propagates information backwards from the data up to the proposal variables. But gradients, and the chain rule, are only defined for continuous variables. Is there a corresponding structure for discrete choices? We now introduce a new nonstandard interpretation based on provenance tracking (PT). In programming language theory, the provenance of a variable is the history of variables and computations that combined to form its value. We use this idea to track fine-grained dependency information between random values and intermediate computations as they combine to form a likelihood. We then use this provenance information to construct good global proposals for discrete variables as part of a novel factored multiple-try MCMC algorithm.

### 4.1   Defining and Implementing Provenance Tracking

Like AD, PT can be implemented with operator overloading. Because provenance information is much coarser than gradient information, the operators in PT objects have a particularly simple form; most program expressions can be covered by considering a few cases. Let $X$ denote the set $\{x_i\}$ of all (not necessarily random) variables in a program. Let $R(x) \subset X$ define the provenance of a variable $x$. Given $R(x)$, the provenance of expressions involving $x$ can be computed by breaking

down expressions into a sequence of unary operations, binary operations, and function applications. Constants have empty provenances.

Let $x$ and $y$ be expressions in the program (consisting of an arbitrary mix of variables, constants, functions and operators). For a binary operation $x \odot y$, the provenance $R(x \odot y)$ of the result is defined to be $R(x \odot y) = R(x) \cup R(y)$. Similarly, for a unary operation, the provenance $R(\odot x) = R(x)$. For assignments, $x = y \Rightarrow R(x) = R(y)$. For a function, $R(f(x, y, ...))$ may be computed by examining the expressions within $f$; a worst-case approximation is $R(f(x, y, ...)) = R(x) \cup R(y) \cdots$. A few special cases are also worth noting. Strictly speaking, the previous rules track a superset of provenance information because some functions and operations are constant for certain inputs. In the case of multiplication, $x * 0 = 0$, so $R(x * 0) = \{\}$. Accounting for this gives tighter provenances, implying, for example, that special considerations apply to sparse linear algebra.

In the case of probabilistic programming, recall that random variables (or ERPs) are represented as stochastic functions $f_i$ that accept parameters $\theta_i$. Whenever a random variable is conditioned, the output of the corresponding $f_i$ is fixed; thus, while the *likelihood* of a particular output of $f_i$ depends on $\theta_i$, the *specific output* of $f_i$ does not. For the purposes inference, therefore, $R(f_i(\theta_i)) = \{\}$.

### 4.2 Using Provenance Tracking as Part of Inference

Provenance information could be used in many ways. Here, we illustrate one use: to help construct good block proposals for MH inference. Our basic idea is to construct a good global proposal by starting with a random global proposal (which is unlikely to be good) and then inhibiting the bad parts. We do this by allowing each element of the likelihood to "vote" on which proposals seemed good. This can be considered a factored version of a multiple-try MCMC algorithm [16].

The algorithm is shown in Fig. 3. Let $x^O$ be the starting state. In step (2), we propose a new state $x^{O'}$. This new state changes many ERPs at once, and is unlikely to be good (for the proof, we require that $x_i^{O'} \neq x_i^O$ for all $i$). In step (3), we accept or reject each element of the proposal based on a function $\alpha$. Our choice of $\alpha$ (Fig. 3, left) uses PT, as we explain below. In step (4) we construct a new proposal $x^M$ by "mixing" two states: we set the variables in the accepted set $A$ to the values of $x_i^{O'}$, and we leave the variables in the rejected set $R$ at their original values in $x^O$. In steps (5-6) we compute the forward probabilities. In steps (7-8) we sample one possible path backwards from $x^M$ to $x^O$, with the relevant probabilities. Finally, in step (9) we accept or reject the overall proposal.

We use $\alpha(x^O, x^{O'})$ to allow the likelihood to "vote" in a fine-grained way for which proposals seemed good and which seemed bad. To do this, we compute $p(x^O)$ using PT to track how each $x_i^O$ influences the overall likelihood $p(x^O)$. Let $D(i; x^O)$ denote the "descendants" of variable $x_i^O$, defined as all ERPs whose likelihood $x_i^O$ impacted. We also use PT to compute $p(x^{O'})$, again tracking dependents $D(i; x^{O'})$, and let $D(i)$ be the joint set of ERPs that $x_i$ influences in either state $x^O$ or $x^{O'}$. We then use $D(i)$, $p(x^O)$ and $p(x^{O'})$ to estimate the amount by which each constituent element $x_i^{O'}$ in the proposal changed the likelihood. We assign "credit" to each $i$ *as if it were the only proposal* – that is, we assume that if, for example, the likelihood went up, it was entirely due to the change in $x_i^O$. Of course, the variables' effects are not truly independent; this is a fully-factored approximation to those effects. The final $\alpha$ is shown in Fig. 3 (left), where we define $p(x_{D(i)})$ to be the likelihood of *only* the subset of variables that $x_i$ impacted.

Here, we prove that our algorithm is valid MCMC by following [16] and showing detailed balance. To do this, we must integrate over all possible rejected paths of the negative bits $x_R^{O'}$ and $x_R^{MI}$:

$$
\begin{aligned}
p(x^O)P(x^M|x^O) &= p(x^O) \int_{x_R^{O'}} \int_{x_R^{MI}} Q_A^{O'} Q_R^{O'} P_A^M P_R^M Q_R^{MI} \min\left\{1, \frac{p(x^M)}{p(x^O)} \frac{Q_A^{MI} P_A^{MI} P_R^{MI}}{Q_A^{O'} P_A^M P_R^M}\right\} \\
&= \int_{x_R^{O'}} \int_{x_R^{MI}} Q_R^{O'} Q_R^{MI} \min\left\{p(x^O) Q_A^{O'} P_A^M P_R^M, p(x^M) Q_A^{MI} P_A^{MI} P_R^{MI}\right\} \\
&= p(x^M) \int_{x_R^{O'}} \int_{x_R^{MI}} Q_A^{MI} Q_R^{MI} P_A^{MI} P_R^{MI} Q_R^{O'} \min\left\{1, \frac{p(x^O) Q_A^{O'} P_A^M P_R^M}{p(x^M) Q_A^{MI} P_A^{MI} P_R^{MI}}\right\} \\
&= p(x^M)P(x^O|x^M)
\end{aligned}
$$

where the subtlety to the equivalence is that the rejected bits $x_R^{O'}$ and $x_R^{MI}$ have switched roles. $\square$

**Alg. 3: Factored Multiple-Try MH**

1: Begin in state $x^O$. Assume it is composed of individual ERPs $x^O = \left\{x_1^O, \cdots, x_k^O\right\}$.

2: Propose a new state for many ERPs. For $i = 1, \cdots, k$, propose $x_i^{O'} \sim Q(x^{O'}|x^O)$ s.t. $x_i^{O'} \neq x_i^O$.

3: Decide to accept or reject each element of $x^{O'}$. This test can depend arbitrarily on $x^O$ and $x^{O'}$, but must decide for each ERP independently; let $\alpha_i(x^O, x^{O'})$ be the probability of accepting $x_i^{O'}$. Let $A$ be the set of indices of accepted proposals, and $R$ the set of rejected ones.

4: Construct a new state, $x^M = \left\{x_i^{O'} : i \in A\right\} \bigcup \left\{x_j^O : j \in R\right\}$. This new state mixes new values for the ERPs from the accepted set $A$ and old values for the ERPs in the rejected set $R$.

5: Let $P_A^M = \prod_{i \in A} \alpha_i(x^O, x^{O'})$ be the probability of accepting the ERPs in $A$, and let $P_R^M = \prod_{j \in R}(1 - \alpha_j(x^O, x^{O'}))$ be the probability of rejecting the ERPs in $R$.

6: Let $Q_A^{O'} = \prod_{i \in A} Q(x_i^{O'}|x^O)$ and $Q_R^{O'} = \prod_{j \in R} Q(x_j^{O'}|x^O)$.

7: Construct a new state $x^{MI}$. Propose new values for all of the rejected ERPs using $x^M$ as the start state, but leave ERPs in the accepted set at their original value. For $j \in R$ let $x_j^{MI} \sim Q(\cdot|x^M)$. Then, $x^{MI} = \left\{x_i^O : i \in A\right\} \bigcup \left\{x_j^{MI} : j \in R\right\}$.

8: Let $P_A^{MI} = \prod_{i \in A} \alpha_i(x^M, x^{MI})$, and let $P_R^{MI} = \prod_{j \in R}(1 - \alpha_j(x^M, x^{MI}))$.

9: Accept $x^M$ with probability $\min\left\{1, (p(x^M)Q_A^{MI}P_A^{MI}P_R^{MI})/(p(x^O)Q_A^{O'}P_A^MP_R^M)\right\}$.

---

**Alg. 4: A PT-based Acceptance Test**

1: The PT algorithm implements $\alpha_i(x, x')$.
2: Compute $p(x)$, tracking $D(x_i; x)$
3: Compute $p(x')$, tracking $D(x_i; x')$
4: Let $D(i) = D(x_i; x) \cup D(x_i; x')$
5: Let $\alpha_i(x, x') = \min\left\{1, \frac{p(x_i')p(x_{D(i)}')}{p(x_i)p(x_{D(i)})}\right\}$

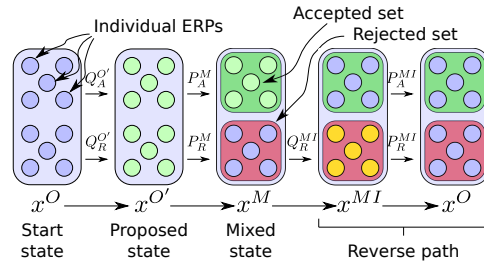

Figure 3: The factored multiple-try MH algorithm (top), the PT-based acceptance test (left) and an illustration of the process of mixing together different elementary proposals (right).

## 4.3 Experiments and Results

We implemented provenance tracking and in Stochastic MATLAB [28] by leveraging MATLAB's object oriented capabilities, which provides full operator overloading. We tested on four tasks: a Bayesian "mesh induction" task, a small QMR problem, probabilistic matrix factorization [23] and an integer-valued variant of PMF. We measured performance by examining likelihood as a function of wallclock time; an important property of the provenance tracking algorithm is that it can help mitigate constant factors affecting inference performance.

**Bayesian mesh induction.** The BMI task is simple: given a prior distribution over meshes and a target image, sample a mesh which, when rendered, looks like the target image. The prior is a Gaussian centered around a "mean mesh," which is a perfect sphere; Gaussian noise is added to each vertex to deform the mesh. The model is shown in Alg. 5. The rendering function is a custom OPENGL renderer implemented as a MEX function. No

**Alg. 5: Bayesian Mesh Induction**

1: function X = bmi( base_mesh )
2:     mesh = base_mesh + randn;
3:     img = render( mesh );
4:     X = img + randn;
5: end;

gradients are available for this renderer, but it is reasonably easy to augment it with provenance information recording vertices of the triangle that were responsible for each pixel. This allows us to make proposals to mesh vertices, while assigning credit based on pixel likelihoods.

Results for this task are shown in Fig. 4 ("Face"). Even though the renderer is quite fast, MCMC with simple proposals fails: after proposing a change to a single variable, it must re-render the image in order to compute the likelihood. In contrast, making large, global proposals is very effective. Fig. 4 (top) shows a sequence of images representing burn-in of the model as it starts from the initial condition and samples its way towards regions of high likelihood. *A video demonstrating the results is available at* `http://www.mit.edu/~wingated/papers/index.html`.

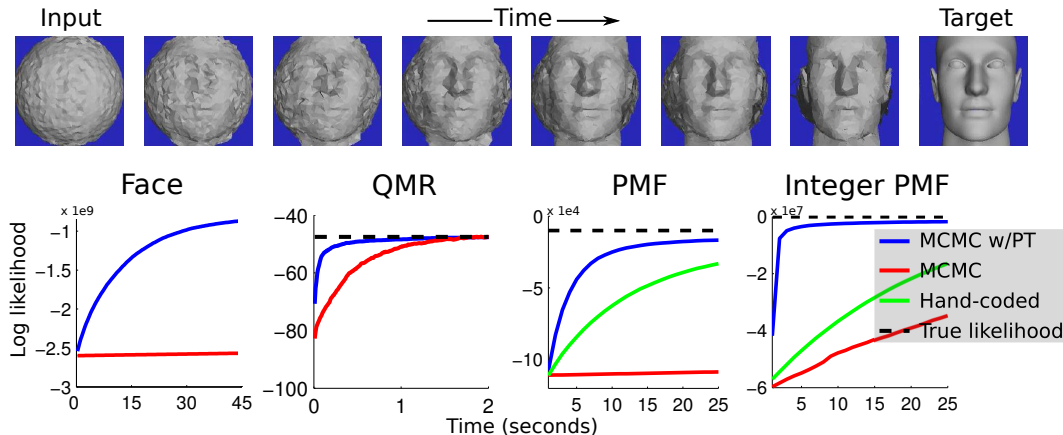

Figure 4: Top: Frames from the face task. Bottom: results on Face, QMR, PMF and Integer PMF.

**QMR.** The QMR model is a bipartite, binary model relating diseases (hidden) to symptoms (observed) using a log-linear noisy-or model. Base rates on diseases can be quite low, so "explaining away" can cause poor mixing. Here, MCMC with provenance tracking is effective: it finds high-likelihood solutions quickly, again outperforming naive MCMC.

**Probabilistic Matrix Factorization.** For the PMF task, we factored a matrix $A \in \mathbb{R}^{1000x1000}$ with 99% sparsity. PMF places a Gaussian prior over two matrices, $U \in \mathbb{R}^{1000x10}$ and $V \in \mathbb{R}^{1000x10}$, for a total of 20,000 parameters. The model assumes that $A_{ij} \sim \mathcal{N}(U_i V_j^T, 1)$. In Fig. 4, we see that MCMC with provenance tracking is able to find regions of much higher likelihood much more quickly than naive MCMC. We also compared to an efficient hand-coded MCMC sampler which is capable of making, scoring and accepting/rejecting about 20,000 proposals per second. Interestingly, MCMC with provenance tracking is more efficient than the hand-coded sampler, presumably because of the economies of scale that come with making global proposals.

**Integer Probabilistic Matrix Factorization.** The Integer PMF task is like ordinary PMF, except that every entry in $U$ and $V$ is constrained to be an integer between 1 and 10. These constraints imply that no gradients exist. Empirically, this does not seem to matter for the efficiency of the algorithm relative to standard MCMC: in Fig. 4 we again see dramatic performance improvements over the baseline Stochastic MATLAB sampler and the hand-coded sampler.

# 5    Conclusions

We have shown how nonstandard interpretations of probabilistic programs can be used to extract structural information about a distribution, and how this information can be used as part of a variety of inference algorithms. The information can take the form of gradients, Hessians, fine-grained dependencies, or bounds. Empirically, we have implemented two such interpretations and demonstrated how this information can be used to find regions of high likelihood quickly, and how it can be used to generate samples with improved statistical properties versus random-walk style MCMC. There are other types of interpretations which could provide additional information. For example, interval arithmetic [22] could be used to provide bounds or as part of adaptive importance sampling.

Each of these interpretations can be used alone or in concert with each other; one of the advantages of the probabilistic programming framework is the clean separation of models and inference algorithms, making it easy to explore combinations of inference algorithms for complex models. More generally, this work begins to illuminate the close connections between probabilistic inference and programming language theory. It is likely that other techniques from compiler design and program analysis could be fruitfully applied to inference problems in probabilistic programs.

## Acknowledgments

DW was supported in part by AFOSR (FA9550-07-1-0075) and by Shell Oil, Inc. NDG was supported in part by ONR (N00014-09-1-0124) and a J. S. McDonnell Foundation Scholar Award. JMS was supported in part by NSF (CCF-0438806), by NRL (N00173-10-1-G023), and by ARL (W911NF-10-2-0060). All views expressed in this paper are the sole responsibility of the authors.

# References

[1] C. Bendtsen and O. Stauning. FADBAD, a flexible C++ package for automatic differentiation. Technical Report IMM–REP–1996–17, Department of Mathematical Modelling, Technical University of Denmark, Lyngby, Denmark, Aug. 1996.

[2] C. H. Bischof, A. Carle, G. F. Corliss, A. Griewank, and P. D. Hovland. ADIFOR: Generating derivative codes from Fortran programs. *Scientific Programming*, 1(1):11–29, 1992.

[3] G. Corliss, C. Faure, A. Griewank, L. Hascoët, and U. Naumann. *Automatic Differentiation: From Simulation to Optimization*. Springer-Verlag, New York, NY, 2001.

[4] J. Eisner, E. Goldlust, and N. A. Smith. Compiling comp ling: Weighted dynamic programming and the Dyna language. In *Proceedings of Human Language Technology Conference and Conference on Empirical Methods in Natural Language Processing (HLT-EMNLP)*, pages 281–290, Vancouver, October 2005.

[5] M. Girolami and B. Calderhead. Riemann manifold Langevin and Hamiltonian Monte Carlo methods. *J. R. Statist. Soc. B*, 73(2):123–214, 2011.

[6] N. Goodman, V. Mansinghka, D. Roy, K. Bonawitz, and J. Tenenbaum. Church: a language for generative models. In *Uncertainty in Artificial Intelligence (UAI)*, 2008.

[7] A. Griewank. *Evaluating Derivatives: Principles and Techniques of Algorithmic Differentiation*. Number 19 in Frontiers in Applied Mathematics. SIAM, 2000.

[8] A. Griewank, D. Juedes, and J. Utke. ADOL-C, a package for the automatic differentiation of algorithms written in C/C++. *ACM Trans. Math. Software*, 22(2):131–167, 1996.

[9] E. Herbst. Gradient and Hessian-based MCMC for DSGE models (job market paper), 2010.

[10] K. Kersting and L. D. Raedt. Bayesian logic programming: Theory and tool. In L. Getoor and B. Taskar, editors, *An Introduction to Statistical Relational Learning*. MIT Press, 2007.

[11] O. Kiselyov and C. Shan. Embedded probabilistic programming. In *Domain-Specific Languages*, pages 360–384, 2009.

[12] Y. LeCun and L. Bottou. Lush reference manual. Technical report, 2002. URL `http://lush.sourceforge.net`.

[13] B. Milch, B. Marthi, S. Russell, D. Sontag, D. L. Ong, and A. Kolobov. BLOG: Probabilistic models with unknown objects. In *International Joint Conference on Artificial Intelligence (IJCAI)*, pages 1352–1359, 2005.

[14] M. B. Monagan and W. M. Neuenschwander. GRADIENT: Algorithmic differentiation in Maple. In *International Symposium on Symbolic and Algebraic Computation (ISSAC)*, pages 68–76, July 1993.

[15] R. M. Neal. MCMC using Hamiltonian dynamics. In *Handbook of Markov Chain Monte-Carlo (Steve Brooks, Andrew Gelman, Galin Jones and Xiao-Li Meng, Eds.)*, 2010.

[16] S. Pandolfi, F. Bartolucci, and N. Friel. A generalization of the multiple-try metropolis algorithm for bayesian estimation and model selection. In *International Conference on Artificial Intelligence and Statistics (AISTATS)*, 2010.

[17] B. A. Pearlmutter and J. M. Siskind. Lazy multivariate higher-order forward-mode AD. In *Symposium on Principles of Programming Languages (POPL)*, pages 155–160, 2007. doi: 10.1145/1190215.1190242.

[18] A. Pfeffer. IBAL: A probabilistic rational programming language. In *International Joint Conference on Artificial Intelligence (IJCAI)*, pages 733–740. Morgan Kaufmann Publ., 2001.

[19] Y. Qi and T. P. Minka. Hessian-based Markov chain Monte-Carlo algorithms (unpublished manuscript), 2002.

[20] P. J. Rossky, J. D. Doll, and H. L. Friedman. Brownian dynamics as smart monte carlo simulation. *Journal of Chemical Physics*, 69:4628–4633, 1978.

[21] D. E. Rumelhart, G. E. Hinton, and R. J. Williams. Learning representations by back-propagating errors. 323:533–536, 1986.

[22] S. Rump. INTLAB - INTerval LABoratory. In *Developments in Reliable Computing*, pages 77–104. Kluwer Academic Publishers, Dordrecht, 1999.

[23] R. Salakhutdinov and A. Mnih. Probabilistic matrix factorization. In *Neural Information Processing Systems (NIPS)*, 2008.

[24] J. M. Siskind and B. A. Pearlmutter. First-class nonstandard interpretations by opening closures. In *Symposium on Principles of Programming Languages (POPL)*, pages 71–76, 2007. doi: 10.1145/1190216.1190230.

[25] B. Speelpenning. *Compiling Fast Partial Derivatives of Functions Given by Algorithms*. PhD thesis, Department of Computer Science, University of Illinois at Urbana-Champaign, Jan. 1980.

[26] B. Taylor. *Methodus Incrementorum Directa et Inversa*. London, 1715.

[27] R. E. Wengert. A simple automatic derivative evaluation program. *Commun. ACM*, 7(8):463–464, 1964.

[28] D. Wingate, A. Stuhlmueller, and N. D. Goodman. Lightweight implementations of probabilistic programming languages via transformational compilation. In *International Conference on Artificial Intelligence and Statistics (AISTATS)*, 2011.

